# Eliciting Categorical Data for Optimal Aggregation

**Chien-Ju Ho**
Cornell University
ch624@cornell.edu

**Rafael Frongillo**
CU Boulder
raf@colorado.edu

**Yiling Chen**
Harvard University
yiling@seas.harvard.edu

## Abstract

Models for collecting and aggregating categorical data on crowdsourcing platforms typically fall into two broad categories: those assuming agents honest and consistent but with heterogeneous error rates, and those assuming agents strategic and seek to maximize their expected reward. The former often leads to tractable aggregation of elicited data, while the latter usually focuses on optimal elicitation and does not consider aggregation. In this paper, we develop a Bayesian model, wherein agents have differing quality of information, but also respond to incentives. Our model generalizes both categories and enables the joint exploration of optimal elicitation and aggregation. This model enables our exploration, both analytically and experimentally, of optimal aggregation of categorical data and optimal multiple-choice interface design.

## 1 Introduction

We study the general problem of eliciting and aggregating information for categorical questions. For example, when posing a classification task to crowd workers who may have heterogeneous skills or amount of information about the underlying true label, the principle wants to elicit workers' private information and aggregate it in a way to maximize the probability that the aggregated information correctly predicts the underlying true label.

Ideally, in order to maximize the probability of correctly predicting the ground truth, the principal would want to elicit agents' full information by asking agents for their entire belief in the form of a probability distribution over labels. However, this is not always practical, e.g., agents might not be able to accurately differentiate 92% and 93%. In practice, the principal is often constrained to elicit agents' information via a multiple-choice interface, which discretizes agents' continuous belief into finite partitions. An example of such an interface is illustrated in Figure 1. Moreover, disregard of whether full or partial information about agents' beliefs is elicited, aggregating the information into a single belief or answer is often done in an ad hoc fashion (e.g. majority voting for simple multiple-choice questions).

In this work, we explore the joint problem of eliciting and aggregating information for categorical data, with a particular focus on how to design the multiple-choice interface, i.e. how to discretize agents' belief space to form discrete choices. The goal is to maximize the probability of correctly predicting the ground truth while incentivizing agents to truthfully report their beliefs. This problem is challenging. Changing the interface not only changes which agent beliefs lead to which responses, but also influences how to optimally aggregate these responses into a single label. Note that we focus on the abstract level of interface design. We explore

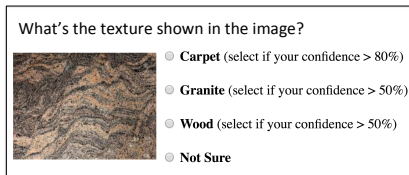

Figure 1: An example of the task interface.

the problem of how to partition agents' belief spaces for optimal aggregations. We do not discuss other behavioral aspects of interface design, such as question framing, layouts, etc

We propose a Bayesian framework, which allows us to achieve our goal in three interleaving steps. First, we constrain our attention to interfaces which admit economically robust payment functions, that is, where agents seeking to maximize their expected payment select the answer that corresponds to their belief. Second, given an interface, we develop a principled way of aggregating information elicited through it, to obtain the maximum a posteriori (MAP) estimator. Third, given the constraints on interfaces (e.g., only binary choice question is allowed) and aggregation methods, we can then choose the optimal interface, which leads to the highest prediction accuracy after both elicitation and aggregation. (Note that if there are no constraints, eliciting full information is always optimal.)

Using theoretical analysis, simulations, and experiments, we provide answers to several interesting questions. Our main results are summarized as follows:

- If the principal can elicit agents' entire belief distributions, our framework can achieve optimal aggregation, in the sense that the principal can make predictions as if she has observed the private information of all agents (Section 4.1). This resolves the open problem of optimal aggregation for categorical data that was considered impossible to achieve in [1].

- For the binary-choice interface design question, we explore the design of optimal interfaces for small and large numbers of agents (Section 4.2). We conduct human-subject experiments on Amazon's Mechanical Turk and demonstrate that our optimal binary-choice interface leads to better prediction accuracy than a natural baseline interface (Section 5.3).

- Our framework gives a simple principled way of aggregating data from arbitrary interfaces (Section 5.1). Applied to experimental data from [2] for a particular multiple-choice interface, our aggregation method has better prediction accuracy than their majority voting (Section 5.2).

- For general multiple-choice interfaces, we use synthetic experiments to obtain qualitative insights of the optimal interface. Moreover, our simple (heuristic) aggregation method performs nearly optimally, demonstrating the robustness of our framework (Section 5.1).

## 1.1 Related Work

Eliciting private information from strategic agents has been a central question in economics and other related domains. The focus here is often on designing payment rules such that agents are incentivized to truthfully report their information. In this direction, proper scoring rules [3, 4, 5] have long been used for eliciting beliefs about categorical and continuous variables. When the realized value of a random variable will be observed, proper scoring rules have been designed for eliciting either the complete subjective probability distributions of the random variable [3, 4] or some statistical properties of these distributions [6, 7]. When the realized value of a random variable will not be available, a class of peer prediction mechanisms [8, 9] has been designed for truthful elicitation. These mechanisms often use proper scoring rules and leverage on the stochastic relationship of agents' private information about the random variable in a Bayesian setting. However, work in this direction often takes elicitation as an end goal and doesn't offer insights on how to aggregate the elicited information.

Another theme in the existing literature is the development of statistical inference and probabilistic modeling methods for the purpose of aggregating agents' inputs. Assuming a batch of noisy inputs, the EM algorithm [10] can be adopted to learn the skill level of agents and obtain estimates of the best answer [11, 12, 13, 14, 15]. Recently, extensions have been made to also consider task assignment and online task assignment in the context of these probabilistic models of agents [16, 17, 18, 19]. Work under this theme often assumes non-strategic agents who have some error rate and are rewarded with a fixed payment that doesn't depend on their reports.

This paper attempts to achieve both truthful elicitation and principled aggregation of information with strategic agents. The closest work to our paper is [1], which has the same general goal and uses a similar Bayesian model of information. That work achieves optimal aggregation by associating the confidence of an agent's prediction with hyperparameters of a conjugate prior distribution. However, this approach leaves optimal aggregation for categorical data as an open question, which we resolve.

Moreover, our model allows us to elicit confidence about an answer over a coarsened report space (e.g. a partition of the probability simplex) and to reason about optimal coarsening for the purpose

of aggregation. In comparison, [2] also elicit quantified confidence on reported labels in their mechanism. Their mechanism is designed to incentivize agents to truthfully report the label that they believe to be correct when their confidence on the report is above a threshold and skip the question when it's below the threshold. Majority voting is then used to aggregate the reported labels. These thresholds provide a coarsened report space for eliciting confidence, and thus are well modeled by our approach. However, in that work the thresholds are given a priori, and moreover, the elicited confidence is not used in aggregation. These are both holes which our approach fill; in Section 5, we demonstrate how to *derive* optimal thresholds, and aggregation policies, which depend critically on the prior distribution and the number of agents.

## 2 Bayesian Model

In our model, the principal would like to get information about a categorical question (e.g., predicting who will win the presidential election, or identifying whether there is a cancer cell in a picture of cells) from $m$ agents. Each question has a finite number of possible answers $\mathcal{X}$, $|\mathcal{X}| = k$. The ground truth (correct answer) $\Theta$ is drawn from a prior distribution $p(\theta)$, with realized value $\theta \in \mathcal{X}$. This prior distribution is common knowledge to the principal and the agents. We use $\theta^*$ to denote the unknown, realized ground truth.

Agents have heterogeneous levels of knowledge or abilities on the question that are unknown to the principal. To model agents' abilities, we assume each agent has observed independent noisy samples related to the ground truth. Hence, each agent's ability can be expressed as the number of noisy samples she has observed. The number of samples observed can be different across agents and is unknown to the principal. Formally, given the ground truth $\theta^*$, each noisy sample $X$, with $x \in \mathcal{X}$, is i.i.d. drawn according to the distribution $p(x|\theta^*)$. [1]

In this paper, we focus our discussion on the symmetric noise distribution, defined as

$$p(x|\theta) = (1 - \epsilon)\mathbb{1}\{\theta = x\} + \epsilon \cdot 1/k.$$

This noise distribution is common knowledge to the principal and the agents. While the symmetric noise distribution may appear restrictive, it is indeed quite general. In Appendix C, we discuss how our model covers many scenarios considered in the literature as special cases.

**Beliefs of Agents.** If an agent has observed $n$ noisy samples, $X_1 = x_1, \ldots, X_n = x_n$, her belief is determined by a count vector $\vec{c} = \{c_\theta : \theta \in \mathcal{X}\}$ where $c_\theta = \sum_{i=1}^n \mathbb{1}\{x_i = \theta\}$ is the number of sample $\theta$ that the agent has observed. According to Bayes' rule, we write her posterior belief on $\Theta$ as $p(\theta|x_1, \ldots, x_n)$, which can be expressed as

$$p(\theta|x_1, \ldots, x_n) = \frac{\prod_{j=1}^n p(x_j|\theta)p(\theta)}{p(x_1, \ldots, x_n)} = \frac{\alpha^{c_\theta} \beta^{n-c_\theta} p(\theta)}{\sum_{\theta' \in \mathcal{X}} \alpha^{c_{\theta'}} \beta^{n-c_{\theta'}} p(\theta')},$$

where $\alpha = 1 - \epsilon + \epsilon/k$ and $\beta = \epsilon/k$.

In addition to the posterior on $\Theta$, the agent also has an updated belief, called the *posterior predictive distribution (PPD)*, about an independent sample $X$ given observed samples $X_1 = x_1, \ldots, X_n = x_n$. The PPD can be considered as a noisy version of the posterior:

$$p(x|x_1, \ldots, x_n) = \frac{\epsilon}{k} + (1 - \epsilon)p(\Theta = x|x_1, \ldots, x_n)$$

In fact, in our setting the PPD and posterior are in one-to-one correspondence, so while our theoretical results focus on the PPD, our experiments will consider the posterior without loss of generality.

**Interface.** An *interface* defines the space of reports the principal can elicit from agents. The reports elicited via the interface naturally partition agents' beliefs, a $k$-dimensional probability simplex, into a (potentially infinite) number of cells, which each correspond to a coarsened version of agents' PPD. Formally, each interface consists of a report space $\mathcal{R}$ and a partition $\mathcal{D} = \{D_r \subseteq \Delta_k\}_{r \in \mathcal{R}}$, with each cell $D_r$ corresponding to a report $r$ and $\bigcup_{r \in \mathcal{R}} D_r = \Delta_k$.[2] In this paper, we sometime use only $\mathcal{R}$ or $\mathcal{D}$ to represent an interface.

In this paper, we focus on the abstract level of the interface design. We explore the problem of how to partition agents' belief spaces for optimal aggregations. We do not discuss other aspects of interface design, such as question framing, layouts, etc. In practice there are often pre-specified constraints on the design of interfaces, e.g., the principal can only ask agents a multiple-choice question with no more than 2 choices. We explore how to optimal design interfaces with given constraints.

**Objective.** The goal of the principal is to choose an interface corresponding to a partition $\mathcal{D}$, satisfying some constraints, and an aggregation method $\mathsf{Agg}_{\mathcal{D}}$, to maximize the probability of correctly predicting the ground truth. One very important constraint is that there should exist a payment method for which agents are correctly incentivized to report $r$ if their belief is in $D_r$; see Section 3. We can formulate the goal as the following optimization problem,

$$\max_{(\mathcal{R},\mathcal{D}) \in \mathsf{Interfaces}} \max_{\mathsf{Agg}_{\mathcal{D}}} \Pr[\mathsf{Agg}_{\mathcal{D}}(R_1, \ldots, R_m) = \Theta] \,, \tag{1}$$

where $R_i$ are random variables representing the reports chosen by agents after $\theta^*$ and the samples are drawn.

## 3 Our Mechanism

We assume the principal has access to a single independent noisy sample $X$ drawn from $p(x|\theta^*)$. The principal can then leverage this sample to elicit and aggregate agents' beliefs by adopting techniques in proper scoring rules [3, 5]. This assumption can be satisfied by, for example, allowing the principal to ask for an additional opinion outside of the $m$ agents, or by asking agents multiple questions and only scoring a small random subset for which answers can be obtained separately (often, on the so-called "gold standard set").

Our mechanism can be described as follows. The principal chooses an interface with report space $\mathcal{R}$ and partition $\mathcal{D}$, and a scoring rule $S(r, x)$ for $r \in \mathcal{R}$ and $x \in \mathcal{X}$. The principal then requests a report $r_i \in \mathcal{R}$ for each agent $i \in \{1, \ldots, m\}$, and observes her own sample $X = x$. She then gives a score of $S(r_i, x)$ to agent $i$ and aggregates the reports via a function $\mathsf{Agg}_{\mathcal{D}} : \mathcal{R} \times \cdots \times \mathcal{R} \to \mathcal{X}$. Agents are assumed to be rational and aim to maximize their expected scores. In particular, if an agent $i$ believes $X$ is drawn from some distribution $p$, she will choose to report $r_i \in \mathrm{argmax}_{r \in \mathcal{R}} \mathbb{E}_{X \sim p}[S(r, X)]$.

**Elicitation.** To elicit truthful reports from agents, we adopt techniques from proper scoring rules [3, 5]. A scoring rule is *strictly proper* if reporting one's true belief uniquely maximizes the expected score. For example, a strictly proper score is the logarithmic scoring rule, $S(p, x) = \log p(x)$, where $p(x)$ is the agent's belief of the distribution $x$ is drawn from.

In our setting, we utilize the requester's additional sample $p(x|\theta^*)$ to elicit agents' PPDs $p(x|x_1, \ldots, x_n)$. If the report space $\mathcal{R} = \Delta_k$, we can simply use any strictly proper scoring rules, such as the logarithmic scoring rule, to elicit truthful reports. If the set of report space $\mathcal{R}$ is finite, we must specify what it means to be truthful. The partition $\mathcal{D}$ defined in the interface is a way of codifying this relationship: a scoring rule is truthful with respect to a partition if report $r$ is optimal whenever an agent's belief lies in cell $D_r$.[3]

**Definition 1.** $S(r, x)$ *is truthful with respect to* $\mathcal{D}$ *if for all* $r \in \mathcal{R}$ *and all* $p \in \Delta_k$ *we have*

$$p \in D_r \iff \forall r' \neq r \; \mathbb{E}_p S(r, X) \geq \mathbb{E}_p S(r', X) \,.$$

Several natural questions arise from this definition. For which partitions $\mathcal{D}$ can we devise such truthful scores? And if we have such a partition, what are all the scores which are truthful for it? As it happens, these questions have been answered in the field of *property elicitation* [20, 21], with the verdict that there exist truthful scores for $\mathcal{D}$ if and only if $\mathcal{D}$ forms a *power diagram*, a type of weighted Voronoi diagram [22].

Thus, when we consider the problem of designing the interface for a crowdsourcing task, if we want to have robust economic incentives, we must confine ourselves to interfaces which induce power

diagrams on the set of agent beliefs. In this paper, we focus on two classes of power diagrams: *threshold partitions*, where the membership $p \in D_r$ can be decided by comparisons of the form $t_1 \leq p_\theta \leq t_2$, and *shadow partitions*, where $p \in D_r \iff r = \operatorname{argmax}_x p(x) - p^*(x)$ for some reference distribution $p^*$. Threshold partitions cover those from [2], and shadow partitions are inspired by the Shadowing Method from peer prediction [23].

**Aggregation.** The goal of the principal is to aggregate the agents' reports into a single prediction which maximizes the probability of correctly predicting the ground truth.

More formally, let us assume that the principal obtains reports $r^1, \ldots, r^m$ from $m$ agents such that the belief $p^i$ of agent $i$ lies in $D^i := D_{r^i}$. In order to maximize the probability of correct predictions, the principal aggregates the reports by calculating the posterior $p(\theta|D^1, \ldots, D^m)$ for all $\theta$ and making the prediction $\hat{\theta}$ that maximizes the posterior.

$$\hat{\theta} = \operatorname*{argmax}_\theta p(\theta|D^1, \ldots, D^m) = \operatorname*{argmax}_\theta \left( \prod_{i=1}^m p(D^i|\theta) \right) p(\theta) \,,$$

where $p(D^i|\theta)$ is the probability that the PPD of agent $i$ falls within $D^i$ giving the ground truth to be $\theta$. To calculate $p(D|\theta)$, we assume agents' abilities, represented by the number of samples, are drawn from a distribution $p(n)$. We assume $p(n)$ is known to the principal. This assumption can be satisfied if the principal is familiar with the market and has knowledge of agents' skill distribution. Empirically, in our simulation, the optimal interface is robust to the choice of this distribution.

$$p(D|\theta) = \sum_n \left( \sum_{x_1..x_n : p(\theta|x_1..x_n) \in D} p(x_1..x_n|\theta) \right) p(n) = \sum_n \left( \sum_{\vec{c}:p(\theta|\vec{c}) \in D} \binom{n}{\vec{c}} \alpha^{c_\theta} \beta^{n-c_\theta} \right) \frac{p(n)}{Z(n)},$$

with $Z(n) = \sum_{\vec{c}} \binom{n}{\vec{c}} \alpha^{c_1} \beta^{n-c_1}$ and $\binom{n}{\vec{c}} = n!/(\prod_i c_i!)$, where $c_i$ is the $i$-th component of $\vec{c}$.

**Interface Design.** Let $P(\mathcal{D})$ be the probability of correctly predicting the ground truth given partition $\mathcal{D}$, assuming the best possible aggregation policy. The expectation is taken over which cell $D^i \in \mathcal{D}$ agent $i$ reports for $m$ agents.

$$P(\mathcal{D}) = \sum_{D^1, \ldots, D^m} \max_\theta p(\theta|D^1, \ldots, D^m) p(D^1, \ldots, D^m)$$

$$= \sum_{D^1, \ldots, D^m} \max_\theta \left( \prod_{i=1}^m p(D^i|\theta) \right) p(\theta) \,.$$

The optimal interface design problem is to find an interface with partition $\mathcal{D}$ within the set of feasible interfaces such that in expectation, $P(\mathcal{D})$ is maximized.

## 4 Theoretical Analysis

In this section, we analyze two settings to illustrate what our mechanism can achieve. We first consider the setting in which the principal can elicit full belief distributions from agents. We show that our mechanism can obtain optimal aggregation, in the sense that the principal can make prediction as if she has observed all the private signals observed by all workers. In the second setting, we consider a common setting with binary signals and binary cells (e.g., binary classification tasks with two-option interface). We demonstrate how to choose the optimal interface when we aim to collect data from one single agent and when we aim to collect data from a large number of agents.

### 4.1 Collecting Full Distribution

Consider the setting in which the allowed reports are full distributions over labels. We show that in this setting, the principal can achieve *optimal* aggregation. Formally, the interface consists of a report space $\mathcal{R} = \Delta_k \subset [0, 1]^k$, which is the $k$-dimensional probability simplex, corresponding to beliefs about the principal's sample $X$ given the observed samples of an agent. The aggregation is *optimal* if the principal can obtain *global PPD*.

**Definition 2** ([1]). *Let $S$ be the set of all samples observed by agents. Given the prior $p(\theta)$ and data $S$ distributed among the agents, the* global PPD *is given by $p(x|S)$.*

In general, as noted in [1], computing the global PPD requires access to agents' actual samples, or at least their counts, whereas the principal can at most elicit the PPD. In that work, it is therefore considered impossible for the principal to leverage a single sample to obtain the global PPD for a categorical question, as there does not exist a unique mapping from PPDs to sample counts. While our setting differs from that paper, we intuitively resolve this impossibility by finding a non-trivial unique mapping between the *differences of sample counts* and PPDs.

**Lemma 1.** *Fix $\theta_0 \in \mathcal{X}$ and let $\mathrm{diff}^i \in \mathbb{Z}^{k-1}$ be the vector $\mathrm{diff}_\theta^i = c_{\theta_0}^i - c_\theta^i$ encoding the differences in the number of samples of $\theta$ and $\theta_0$ that agent $i$ has observed. There exists an unique mapping between $\mathrm{diff}^i$ and the PPD of agent $i$.*

With Lemma 1 in hand, assuming the principal can obtain the full PPD from each agent, she can now compute the global PPD: she simply converts each agents' PPD into a sample count difference, sums these differences, and finally converts the total differences into the global PPD.

**Theorem 2.** *Given the PPDs of all agents, the principal can obtain the global PPD.*

### 4.2  Interface Design in Binary Settings

To gain the intuition about optimal interface design, we examine a simple setting with binary signal $\mathcal{X} = \{0, 1\}$ and a partitions with only two cells. To simplify the discussion, we also assume all agents have observed exactly $n$ samples. In this setting, each partition can be determined by a single parameter, the threshold $p_T$; its cells indicate whether the agent believes the probability of the principal's sample $X$ to be 0 is larger than $p_T$ or not. Note that we can also write the threshold as $T$, the number of samples that the agent observes to be signal 0. Membership in the two cells indicates whether or not the agents observes more than $T$ samples with signal 0.

We first give the result when there is only one agent. [4]

**Lemma 3.** *In the binary-signal and two-cell setting, if the number of agents is one, the optimal partition has threshold $p_{T^*} = 1/2$.*

If the number of agents is large, we numerically solve for the optimal partition with a wide range of parameters. We find that the optimal partition is to set the threshold such that agents' posterior belief on the ground truth is the same as the prior. This is equivalent to asking agents whether they observe more samples with signal 0 or with signal 1. Please see Appendix B and H for more discussion.

The above arguments suggest that when the principal plans to collect data from multiple agents for datasets with asymmetric priors (e.g., identifying anomaly images from a big dataset), adopting our interface would lead to better aggregation than traditional interface do. We have evaluated this claim in real-world experiments in Section 5.3.

## 5  Experiments

To confirm our theoretical results and test our model, we turn to experimental results. In our synthetic experiments, we simply explore what the model tells us about optimal partitions and how they behave as a function of the model, giving us qualitative insights into interface design. We also introduce a *heuristic aggregation* method, which allows our results to be easily applied in practice. In addition to validating our heuristics numerically, we show that they lead to real improvements over simple majority voting by re-aggregating some data from previous work [2]. Finally, we perform our own experiments for a binary signal task and show that the optimal mechanism under the model, coupled with heuristic aggregation, significantly outperforms the baseline.

### 5.1  Synthetic Experiments

From our theoretical results, we expect that in the binary setting, the boundary of the optimal partition should be roughly uniform for small numbers of agents and quickly approach the prior as the number of agents per task increases. In the Appendix, we confirm this numerically. Figure 2 extends this intuition to the 3-signal case, where the optimal reference point $p^*$ for a shadow partition closely tracks the prior. Figure 2 also gives insight into the design of threshold partitions, showing

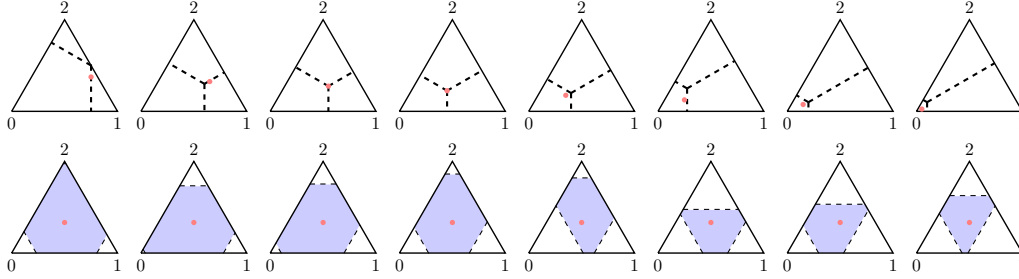

Figure 2: Optimal interfaces as a function of the model; the prior is shown in each as a red dot. Each triangle represents the probability simplex on three signals (0,1,2), and the cells (set of posteriors) of the partition defined by the interface are delineated by dashed lines. **Top:** the optimal shadow partition for three agents. Here the reference distribution $p^*$ is close to the prior, but often slightly toward uniform as suggested by the behavior in the binary case (Section 4.2); for larger numbers of agents this point in fact matches the prior always. **Bottom:** the optimal threshold partition for increasing values of $\epsilon$. Here as one would expect, the more uncertainty agents have about the true label, the lower the thresholds should be.

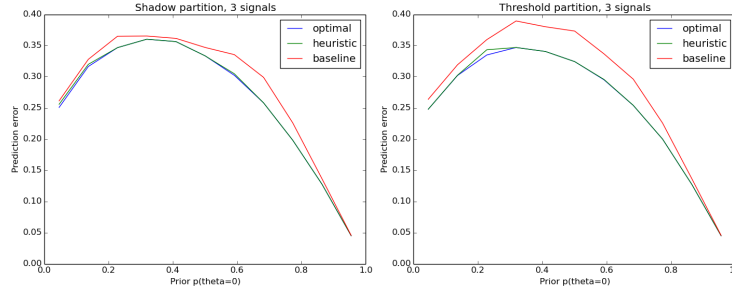

Figure 3: Prediction error according to our model as a function of the prior for (a) the optimal partition with optimal aggregation, (b) the optimal partition with heuristic aggregation, and (c) the naïve partition and aggregation. As we see, the heuristics are nearly optimal, and yield significantly lower error than the baseline.

that the threshold values should decrease as agent uncertainty increases. The Appendix gives other qualitative findings.

The optimal partitions and aggregation policies suggested by our framework are often quite complicated. Thus, to be practical, one would like simple partitions and aggregation methods which perform nearly optimally under our framework. Here we suggest a *heuristic aggregation* (HA) method which is defined for a fixed number of samples $n$: for each cell $D_r$, consider the set of count vectors after which an agent's posterior would lie in $D_r$, and let $c_r$ be the average count vector in this set. Now when agents report $r_1, \ldots, r_m$, simply sum the count vectors and choose $\hat{\theta} = \mathsf{HA}(r_1, \ldots, r_m) = \operatorname{argmax}_\theta p(\theta | c_{r_1} + \ldots + c_{r_m})$. Thus, by simply translating the choice of cell $D_r$ to a representative sample count an agent may have observed, we arrive at a weighted-majority-like aggregation method. This simple method performs quite well in simulations, as Figure 3 shows. It also performs well in practice, as we will see in the next two subsections.

## 5.2 Aggregation Results for Existing Mechanisms

We evaluate our heuristic aggregation method using the dataset collected from existing mechanisms in previous work [2]. Their dataset is collected by asking workers to answer a multi-choice question and select one of the two confidence levels at the same time. We compared our heuristic aggregation (HA) with simple majority voting (Maj) as adopted in their paper. For our heuristics, we used the model with $n = 4$ and $\epsilon = 0.85$ for every case here; this was the simplest model for which every cell in every partition contained at least one possible posterior. Our results are fairly robust to the choice of the model subject to this constraint, however, and often other models perform even better. In Figure 4, we demonstrate the aggregation results for one of their tasks ("National Flags") in the dataset. Although the improvement is relatively small, it is statistically significant for every setting plotted. Our HA outperformed Maj for all of their datasets and for all values of $m$.

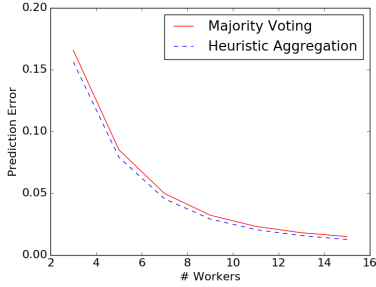

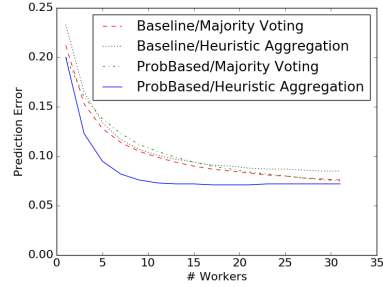

Figure 4: The prediction error of aggregating data collect from existing mechanisms in previous work [2].

Figure 5: The prediction error of aggregating data collected from Amazon Mechanical Turk.

## 5.3 Experiments on Amazon Mechanical Turk

We conducted experiments on Amazon Mechanical Turk (mturk.com) to evaluate our interface design. Our goal was to examine whether workers respond to different interfaces, and whether the interface and aggregation derived from our framework actually leads to better predictions.

**Experiment setup.** In our experiment, workers are asked to label 20 blurred images of textures. We considered an asymmetric prior: 80% of the images were carpet and 20% were granite, and we communicated this to the workers. Upon accepting the task, workers were randomly assigned to one of two treatments: Baseline or ProbBased. Both offered a base payment of 10 cents, but the bonus payments on the 5 randomly chosen "ground truth" images differed between the treatments.

The Baseline treatment is the mostly commonly seen interface in crowdsourcing markets. For each image, the worker is asked to choose from {Carpet, Granite}. She can get a bonus of 4 cents for each correct answer in the ground truth set. In the ProbBased interface, the worker was asked whether she thinks the probability of the image to be Carpet is {more than 80%, no more than 80%}. From Section 4.2, this threshold is optimal when we aim to aggregate information from a potentially large number of agents. To simplify the discussion, we map the two options to {Carpet, Granite} for the rest of this section. For the 5 randomly chosen ground truth images, the worker would get 2 cents for each correct answer of carpet images, and get 8 cents for each correct answer of granite images. We tuned the bonus amount such that the expected bonus for answering all questions correctly is approximately the same for each treatment. One can also easily check that for these bonus amounts, workers maximize their expected bonus by honestly reporting their beliefs.

**Results.** This experiment is completed by 200 workers, 105 in Baseline and 95 in ProbBased. We first observe whether workers' responses differ for different interfaces. In particular, we compare the ratio of workers reporting Granite. As shown in Figure 6 (in Appendix A), our result demonstrates that workers do respond to our interface design and are more likely to choose Granite for all images. The differences are statistically significant ($p < 0.01$). We then examine whether this interface combined with our heuristic aggregation leads to better predictions. We perform majority voting (Maj) for Baseline, and apply our heuristic aggregation (HA) to ProbBased. We choose the simplest model ($n = 1$) for HA though the results are robust for higher $n$. Figure 5 shows that our interface leads to considerably smaller aggregation for different numbers of randomly selected workers. Performing HA for Baseline and Maj for ProbBased both led to higher aggregation errors, which underscores the importance of matching the aggregation to the interface.

## 6    Conclusion

We have developed a Bayesian framework to model the elicitation and aggregation of categorical data, giving a principled way to aggregate information collected from arbitrary interfaces, but also to design the interfaces themselves. Our simulation and experimental results show the benefit of our framework, resulting in significant prediction performance gains over standard interfaces and aggregation methods. Moreover, our theoretical and simulation results give new insights into the design of optimal interfaces, some of which we confirm experimentally. While certainly more experiments are needed to fully validate our methods, we believe our general framework to have value when designing interfaces and aggregation policies for eliciting categorical information.

**Acknowledgments**

We thank the anonymous reviewers for their helpful comments. This research was partially supported by NSF grant CCF-1512964, NSF grant CCF-1301976, and ONR grant N00014-15-1-2335.

## Footnotes

[1] When there is no ambiguity, we use $p(x|\theta^*)$ to represent $p(X = x|\Theta = \theta^*)$ and similar notations for other distributions.

[2] Strictly speaking, we will allow cells to overlap on their boundary; see Section 3 for more discussion.

[3]As mentioned above, strictly speaking, the cells $\{D_r\}_{r \in \mathcal{R}}$ do not form a partition because their boundaries overlap. This is necessary: for any (nontrivial) finite-report mechanism, there exist distributions for which the agent is indifferent between two or more reports. Fortunately, the set of all such distributions has Lebesgue measure 0 in the simplex, so these boundaries do not affect our analysis.

[4]Our result can be generalized to $k$ signals and one agent. See Lemma 4 in Appendix G.

# References

[1] R. M. Frongillo, Y. Chen, and I. Kash. Elicitation for aggregation. In *The Twenty-Ninth AAAI Conference on Artificial Intelligence*, 2015.

[2] N. B. Shah and D. Zhou. Double or Nothing: Multiplicative Incentive Mechanisms for Crowdsourcing. In *Neural Information Processing Systems*, NIPS '15, 2015.

[3] Glenn W. Brier. Verification of forecasts expressed in terms of probability. *Monthly Weather Review*, 78(1):1–3, 1950.

[4] L. J. Savage. Elicitation of personal probabilities and expectations. *Journal of the American Statistical Association*, 66(336):783–801, 1971.

[5] T. Gneiting and A. E. Raftery. Strictly Proper Scoring Rules, Prediction, and Estimation. *Journal of the American Statistical Association*, 102(477):359–378, 2007.

[6] N.S. Lambert, D.M. Pennock, and Y. Shoham. Eliciting properties of probability distributions. In *Proceedings of the 9th ACM Conference on Electronic Commerce*, EC '08, pages 129–138. ACM, 2008.

[7] R. Frongillo and I. Kash. Vector-Valued Property Elicitation. In *Proceedings of the 28th Conference on Learning Theory*, pages 1–18, 2015.

[8] N. Miller, P. Resnick, and R. Zeckhauser. Eliciting informative feedback: The peer-prediction method. *Management Science*, 51(9):1359–1373, 2005.

[9] D. Prelec. A bayesian truth serum for subjective data. *Science*, 306(5695):462–466, 2004.

[10] A. P. Dempster, N. M. Laird, and D. B. Rubin. Maximum likelihood from incomplete data via the EM algorithm. *Journal of the Royal Statistical Society: Series B*, 39:1–38, 1977.

[11] V. Raykar, S. Yu, L. Zhao, G. Valadez, C. Florin, L. Bogoni, and L. Moy. Learning from crowds. *Journal of Machine Learning Research*, 11:1297–1322, 2010.

[12] S. R. Cholleti, S. A. Goldman, A. Blum, D. G. Politte, and S. Don. Veritas: Combining expert opinions without labeled data. In *Proceedings 20th IEEE international Conference on Tools with Artificial intelligence*, 2008.

[13] R. Jin and Z. Ghahramani. Learning with multiple labels. In *Advances in Neural Information Processing Systems*, volume 15, pages 897–904, 2003.

[14] J. Whitehill, P. Ruvolo, T. Wu, J. Bergsma, and J. Movellan. Whose vote should count more: Optimal integration of labels from labelers of unknown expertise. In *Advances in Neural Information Processing Systems*, volume 22, pages 2035–2043, 2009.

[15] A. P. Dawid and A. M. Skene. Maximum likelihood estimation of observer error-rates using the EM algorithm. *Applied Statistics*, 28:20–28, 1979.

[16] D. R. Karger, S. Oh, and D. Shah. Iterative learning for reliable crowdsourcing systems. In *The 25th Annual Conference on Neural Information Processing Systems (NIPS)*, 2011.

[17] D. R. Karger, S. Oh, and D. Shah. Budget-optimal crowdsourcing using low-rank matrix approximations. In *Proc. 49th Annual Conference on Communication, Control, and Computing (Allerton)*, 2011.

[18] J. Zou and D. C. Parkes. Get another worker? Active crowdlearning with sequential arrivals. In *Proceedings of the Workshop on Machine Learning in Human Computation and Crowdsourcing*, 2012.

[19] C. Ho, S. Jabbari, and J. W. Vaughan. Adaptive task assignment for crowdsourced classification. In *The 30th International Conference on Machine Learning (ICML)*, 2013.

[20] N. Lambert and Y. Shoham. Eliciting truthful answers to multiple-choice questions. In *Proceedings of the Tenth ACM Conference on Electronic Commerce*, EC '09, pages 109–118, 2009.

[21] R. Frongillo and I. Kash. General truthfulness characterizations via convex analysis. In *Web and Internet Economics*, pages 354–370. Springer, 2014.

[22] F. Aurenhammer. Power diagrams: properties, algorithms and applications. *SIAM Journal on Computing*, 16(1):78–96, 1987.

[23] J. Witkowski and D. Parkes. A robust bayesian truth serum for small populations. In *Proceedings of the 26th AAAI Conference on Artificial Intelligence*, AAAI '12, 2012.

[24] V. Sheng, F. Provost, and P. Ipeirotis. Get another label? Improving data quality using multiple, noisy labelers. In *ACM SIGKDD Conferences on Knowledge Discovery and Data Mining (KDD)*, 2008.

[25] P. Ipeirotis, F. Provost, V. Sheng, and J. Wang. Repeated labeling using multiple noisy labelers. *Data Mining and Knowledge Discovery*, 2014.

